# Dynamical segmentation of single trials from population neural data

**Biljana Petreska**
Gatsby Computational Neuroscience Unit
University College London
biljana@gatsby.ucl.ac.uk

**Byron M. Yu**
ECE and BME
Carnegie Mellon University
byronyu@cmu.edu

**John P. Cunningham**
Dept of Engineering
University of Cambridge
jpc74@cam.ac.uk

**Gopal Santhanam, Stephen I. Ryu[†], Krishna V. Shenoy[‡]**
Electrical Engineering
[‡]Bioengineering, Neurobiology and Neurosciences Program
Stanford University
[†]Dept of Neurosurgery, Palo Alto Medical Foundation
{gopals,seoulman,shenoy}@stanford.edu

**Maneesh Sahani**
Gatsby Computational Neuroscience Unit
University College London
maneesh@gatsby.ucl.ac.uk

## Abstract

Simultaneous recordings of many neurons embedded within a recurrently-connected cortical network may provide concurrent views into the dynamical processes of that network, and thus its computational function. In principle, these dynamics might be identified by purely unsupervised, statistical means. Here, we show that a Hidden Switching Linear Dynamical Systems (HSLDS) model—in which multiple linear dynamical laws approximate a nonlinear and potentially non-stationary dynamical process—is able to distinguish different dynamical regimes within single-trial motor cortical activity associated with the preparation and initiation of hand movements. The regimes are identified without reference to behavioural or experimental epochs, but nonetheless transitions between them correlate strongly with external events whose timing may vary from trial to trial. The HSLDS model also performs better than recent comparable models in predicting the firing rate of an isolated neuron based on the firing rates of others, suggesting that it captures more of the "shared variance" of the data. Thus, the method is able to trace the dynamical processes underlying the coordinated evolution of network activity in a way that appears to reflect its computational role.

## 1 Introduction

We are now able to record from hundreds—and very likely soon from thousands—of neurons in vivo. By studying the activity of these neurons in concert we may hope to gain insight not only into the computations performed by specific neurons, but also into the computations performed by the population as a whole. The dynamics of such collective computations can be seen in the coordinated activity of all of the neurons within the local network; although each individual such neuron may reflect this coordinated component only noisily. Thus, we hope to identify the computationally-relevant network dynamics by purely statistical, unsupervised means—capturing the shared evolu-

tion through latent-variable state-space models [1, 2, 3, 4, 5, 6, 7, 8]. The situation is similar to that of a camera operating at the extreme of its light sensitivity. A single pixel conveys very little information about an object in the scene, both due to thermal and shot noise and due to the ambiguity of the single-channel signal. However, by looking at all of the noisy pixels simultaneously and exploiting knowledge about the structure of natural scenes, the task of extracting the object becomes feasible. In a similar way, noisy data from many neurons participating in a local network computation needs to be combined with the learned structure of that computation—embodied by a suitable statistical model—to reveal the progression of the computation.

Neural spiking activity is usually analysed by averaging across multiple experimental trials, to obtain a smooth estimate of the underlying firing rates [2, 3, 4, 5]. However, even under carefully controlled experimental conditions, the animal's behavior may vary from trial-to-trial. Reaction time in motor or decision-making tasks for example, reflects internal processes that can last for measurably different periods of time. In these cases traditional methods are challenging to apply, as there is no obvious way of aligning the data from different trials. It is thus essential to develop methods for the analysis of neural data that can account for the timecourse of a neural computation during a single trial. Single-trial methods are also attractive for analysing specific trials in which the subject exhibits erroneous behavior. In the case of a surprisingly long movement preparation time or a wrong decision, it becomes possible to identify the sources of error at the neural level. Furthermore, single-trial methods allow the use of more complex experimental paradigms where the external stimuli can arise at variable times (e.g. variable time delays).

Here, we study a method for the unsupervised identification of the evolution of the network computational state on single trials. Our approach is based on a Hidden Switching Linear Dynamical System (HSLDS) model, in which the coordinated network influence on the population is captured by a low-dimensional latent variable which evolves at each time step according to one of a set of available linear dynamical laws. Similar models have a long history in tracking, speech and, indeed, neural decoding applications [9, 10, 11] where they are variously known as Switching Linear Dynamical System models, Jump Markov models or processes, switching Kalman Filters or Switching Linear Gaussian State Space models [12]. We add the prefix "Hidden" to stress that in our application neither the switching process nor the latent dynamical variable are ever directly observed, and so learning of the parameters of the model is entirely unsupervised—and again, learning in such models has a long history [13]. The details of the HSLDS model, inference and learning are reviewed in Section 2. In our models, the transitions between linear dynamical laws may serve two purposes. First, they may provide a piecewise-linear approximation to a more accurate non-linear dynamical model [14]. Second, they may reflect genuine changes in the dynamics of the local network, perhaps due to changes in the goals of the underlying computation under the control of signals external to the local area. This second role leads to a computational segmentation of individual trials, as we will see below.

We compare the performance of the HSLDS model to Gaussian Processes Factor Analysis (GPFA), a method introduced by [8] which analyses multi-neuron data on a single-trial basis with similar motivation to our own. Instead of explicitly modeling the network computation as a dynamical process, GPFA assumes that the computation evolves smoothly in time. In this sense, GPFA is less restrictive and would perform better if the HSLDS provided a bad model of the real network dynamics. However GPFA assumes that the latent dimensions evolve independently, making GPFA more restrictive than HSLDS in which the latent dimensions can be coupled. Coupling the latent dynamics introduces complex interactions between the latent dimensions, which allows a richer set of behaviors. To validate our HSLDS model against GPFA and a single LDS we will use the cross-prediction measure introduced with GPFA [8] in which the firing rate of each neuron is predicted using only the firing rates of the rest of the neurons; thus the metric measures how well each model captures the shared components of the data. GPFA and cross-prediction are reviewed briefly in Section 3, which also introduces the dataset used; and the cross-prediction performance of the models is compared in Section 4. Having validated the HSLDS approach, we go on to study the dynamical segmentation identified by the model in the rest of Section 4, leading to the conclusions of Section 5.

## 2 Hidden Switching Linear Dynamical Systems

Our goal is to extract the structure of computational dynamics in a cortical network from the recorded firing rates of a subset of neurons in that network. We use a Hidden Switching Linear Dynamical Systems (HSLDS) model to capture the component of those firing rates which is shared by multiple cells, thus exploiting the intuition that network computations should be reflected in coordinated activity across a local population. This will yield a latent low-dimensional subspace of dynamical states embedded within the space of noisy measured firing rates, along with a model of the dynamics within that latent space. The dynamics of the HSLDS model combines a number of linear dynamical systems (LDS), each of which capture linear Markovian dynamics using a first-order linear auto-regressive (AR) rule [9, 15]. By combining multiple such rules, the HSLDS model can provide a piecewise linear approximation to nonlinear dynamics, and also capture changes in the dynamics of the local network driven by external influences that presumably reflect task demands. In the model implemented here, transitions between LDS rules themselves form a Markov chain.

Let $x_{:,t} \in \mathbb{R}_{p \times 1}$ be the low-dimensional computational state that we wish to estimate. This latent computational state reflects the network-level computation performed at timepoint $t$ that gives rise to the observed spiking activity $y_{:,t} \in \mathbb{R}_{q \times 1}$. Note that the dimensionality of the computational state $p$ is lower than the dimensionality of the recorded neural data $q$ which corresponds to the number of recorded neurons. The evolution of the computational state $x_{:,t}$ is given by

$$x_{:,t}|x_{:,t-1}, s_t \sim \mathcal{N}(A_{s_t} x_{:,t-1}, K_{s_t}) \tag{1}$$

where $\mathcal{N}(\mu, \Sigma)$ denotes a Gaussian distribution with mean $\mu$ and covariance $\Sigma$. The linear dynamical matrices $A_{s_t} \in \mathbb{R}_{p \times p}$ and innovations covariance matrices $K_{s_t} \in \mathbb{R}_{p \times p}$ are parameters of the model and need to be learned. These matrices are indexed by a switch variable $s_t \in \{1, ..., S\}$ such that different $A_{s_t}$ and $K_{s_t}$ need to be learned for each of the $S$ possible linear dynamical systems. If the dependencies on $s_t$ are removed, Eq. 1 defines a single LDS.

The switch variable $s_t$ specifies which linear dynamical law guides the evolution of the latent state $x_{:,t}$ at timepoint $t$ and as such provides a piecewise approximation to the nonlinear dynamics with which $x_{:,t}$ may evolve. The variable $s_t$ itself is drawn from a Markov transition matrix $M$ learned from the data:

$$s_t \sim \mathsf{Discrete}(M_{:,s_{t-1}})$$

As mentioned above, the observed neural activity $y_{:,t} \in \mathbb{R}_{q \times 1}$ is generated by the latent dynamics and denotes the spike counts (Gaussianised as described below) of $q$ simultaneously recorded neurons at timepoints $t \in \{1, ..., T\}$. The observations $y_{:,t}$ are related to the latent computational states $x_{:,t}$ through a linear-Gaussian relationship:

$$y_{:,t}|x_{:,t} \sim \mathcal{N}(Cx_{:,t} + d, R).$$

where the observation matrix $C \in \mathbb{R}_{q \times p}$, offset $d \in \mathbb{R}_{q \times 1}$, and covariance matrix $R \in \mathbb{R}_{q \times q}$ are model parameters that need to be learned. We force $R$ to be diagonal and to keep track only of the independent noise variances. This means that the firing rates of different neurons are independent conditioned on the latent dynamics, compelling the shared variance to live only in the latent space. Note that different neurons can have different independent noise variances. We use a Gaussian relationship instead of a point-process likelihood model for computational tractability. Finally, the observation dynamics do not depend on which linear dynamical system is used (i.e., are independent of $s_t$). A graphical model of the particular HSLDS instance we have used is shown in Figure 2.

Inference and learning in the model are performed by approximate Expectation Maximisation (EM). Inference (or the E-step) requires finding appropriate expected sufficient statistics under the distributions of the computational latent state and switch variable at each point in time given the observed neural data $p(x_{1:T}, s_{1:T}|y_{1:T})$. Inference in the HSLDS is computationally intractable because of the following exponential complexity. At the initial timepoint, $s_0$ can take one of $S$ discrete values. At the next timepoint, each of the $S$ possible latent states can again evolve according to $S$ different linear dynamical laws, such that at timepoint $t$ we need to keep track of $S^t$ possible solutions. To avoid

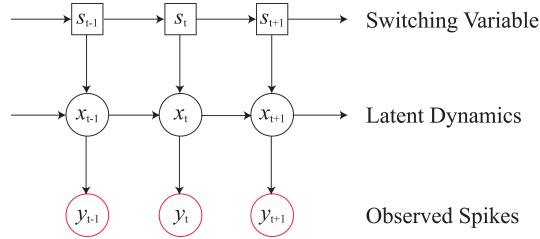

Figure 1: Graphical model of the HSLDS. The first layer corresponds to the discrete switch variable that dictates which of the $S$ available linear dynamical systems (LDSs) will guide the latent dynamics shown in the second layer. The latent dynamics evolves as a linear dynamical system at timepoint $t$ and presumably captures relevant aspects of the computation performed at the level of the recorded neural network. The relationship between the latent dynamics and neural data (third layer) is again linear-Gaussian, such that each computational state is associated to a specific denoised firing pattern. The dimensionality of the latent dynamics $x$ is lower than that of the observations $y$ (equivalent to the number of recorded neurons), meaning that $x$ extracts relevant features reflected in the shared variance of $y$. Note that there are no connections between $x_{t-1}$ and $s_t$, nor $s_t$ and $y$.

this exponential scaling, we use an approximate inference algorithm based on Assumed Density Filtering [16, 17, 18] and Assumed Density Smoothing [19]. The algorithm comprises a single forward pass that estimates the filtered posterior distribution $p(x_t, s_t | y_{1:t})$ and a single backward pass that estimates the smoothed posterior distribution $p(x_t, s_t | y_{1:T})$. The key idea is to approximate these posterior distributions by a simple tractable form such as a single Gaussian. The approximated distribution is then propagated through time conditioned on the new observation. The smoothing step requires an additional simplifying assumption where $p(x_{t+1} | s_t, s_{t+1}, y_{1:T}) \approx p(x_{t+1} | s_{t+1}, y_{1:T})$ as proposed in [19]. It is also possible to use a mixture of a fixed number of Gaussians as the approximating distribution, at the cost of greater computational time. We found that this approach yielded similar results in pilot runs, and thus retained the single-Gaussian approximation.

Learning the model parameters (or the M-step) can be performed using the standard procedure of maximizing the expected joint log-likelihood:

$$\sum_{n=1}^{N} \left\langle \log p(x_{1:T}^n, y_{1:T}^n) \right\rangle_{p^{old}(x^n | y^n)}$$

with respect to the parameters $A_{s_t}$, $K_{s_t}$, $M$, $C$, $d$ and $R$, where the superscript $n$ indexes data from each of $N$ different trials. In practice, the estimated individual variance of particularly low-firing neurons was very low and likely to be incorrectly estimated. Therefore we assumed a Wishart prior on the observation covariance matrix $R$, which resulted in an update rule that adds a fixed parameter $\psi \in \mathbb{R}$ to all of the values at the diagonal. In the analyses below $\psi$ was fixed to the value that gave the best cross-prediction results (see Section 3.2). Finally, the most likely state of the switch variable $s_{1:T}^* = \arg \max_{s_{1:T}} p(s_{1:T} | y_{1:T})$ was estimated using the standard Viterbi algorithm [20], which ensures that the most likely switch variable path is in fact possible in terms of the transitions allowed by $M$.

## 3 Model Comparison and Experimental Data

### 3.1 Gaussian Process Factor Analysis

Below, we compare the performance of the HSLDS model to Gaussian Process Factor Analysis (GPFA), another method for estimating the functional computation of a set of neurons. GPFA is an extension of Factor Analysis that leverages time-label information, introduced in [8]. In this model, the latent dynamics evolve as a Gaussian Process (GP), with a smooth correlation structure between the latent states at different points in time. This combination of FA and the GP prior work together to identify smooth low-dimensional latent trajectories.

Formally, each dimension of the low-dimensional latent states $x_{:,t}$ is indexed by $i \in \{1, ..., p\}$ and defines a separate GP:

$$x_{i,:} \sim \mathcal{N}(0, K_i)$$

where $x_{i,:} \in I\!R_{1 \times T}$ is the trajectory in time of the $i$th latent dimension and $K_i \in I\!R_{T \times T}$ is the $i$th GP smoothing covariance matrix. $K_i$ is set to the commonly-used squared exponential (SE) covariance function as defined in [8].

Whereas HSLDS explicitly models the dynamics of the network computation, GPFA only assumes that the evolution of the computational state is smooth. Thus GPFA is a less restrictive model than HSLDS, but being model-free makes it also less informative of the dynamical rules that underlie the computation. A major advantage of GPFA over HSLDS is that the solution is approximation-free and faster to run.

## 3.2 Cross-prediction performance measure

To compare model goodness-of-fit we adopt the cross-prediction metric of [8]. All of these models attempt to capture the shared variance in the data, and so performance may be measured by how well the activity of one neuron can be predicted using the activity of the rest of the neurons. It is important to measure the cross-prediction error on trials that have not been used for learning the parameters of the model. We arrange the observed neural data in a matrix $Y = [y_{:,1}, ..., y_{:,T}] \in I\!R_{q \times T}$ where each row $y_{j,:}$ represents the activity of neuron $j$ in time. The model cross-prediction for this neuron $j$ is $\hat{y}_{j,:} = E[y_{j,:}|Y_{-j,:}]$ where $Y_{-j,:} \in I\!R_{(q-1) \times T}$ represents all but the $j$th row of $Y$. We first estimate the trajectories in the latent space using all but the $j$th neuron $P(x_{1:p,:}|Y_{-j,:})$ in a set of testing trials. We then project this estimate back to the high-dimensional space to obtain the model cross-prediction $\hat{y}_{j,:}$ using $\hat{y}_{j,t} = C_{j,:} \cdot E[x(:,t)|Y_{-j,:}] + d_j$. The error is computed as the sum-of-squared errors between the model cross-prediction and the observed Gaussianised spike counts across all neurons and timepoints; and we plot the difference between this error (per time bin) and the average temporal variance of the corresponding neuron in the corresponding trial (denoted as Var-MSE).

Note that the performance of difference models can be evaluated as a function of the dimensionality of the latent state. The HSLDS model has two futher free parameters which influence cross-prediction peformance: the number of available LDSs $S$ and the concentration of the Wishart prior $\psi$.

## 3.3 Data

We applied the model to data recorded in the premotor and motor cortices of a rhesus macaque while it performed a delayed center-out reach task. A trial began with the animal touching and looking at an illuminated point at the center of a vertically oriented screen. A target was then illuminated at a distance of 10cm and in one of seven directions (0, 45, 90, 135, 180, 225, 315) away from this central starting point. The target remained visible while the animal prepared but withheld a movement to touch it. After a random delay of between 200 and 700ms, the illumination of the starting point was extinguished, which was the animal's cue (the "go cue") to reach to the target to obtain a reward. Neural activity was recorded from 105 single and multi-units, using a 96-electrode array (Blackrock, Salt Lake City, UT). All active units were included in the analysis without selection based on tuning. The spike-counts were binned at a relatively fine time-scale of 10ms (non-overlapping bins). As in [8], the observations were taken to be the square-roots of these spike counts, a transformation that helps to Gaussianise and stabilise the variance of count data [21].

## 4 Results

We first compare the cross-prediction-derived goodness-of-fit of the HSLDS model to that of the single LDS and GPFA models in section 4.1. We find that HSLDS provides a better model of the shared component of the recorded data than do the two other methods. We then study the dynamical segmentation found by the HSLDS model, first by looking at a typical example (section 4.2) and then by correlating dynamical switches to behavioural events (section 4.3). We show that the latent

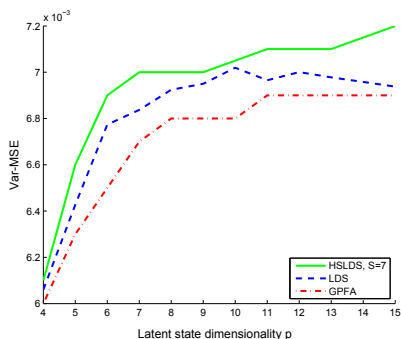

Figure 2: Performance of the HSLDS (green solid line), LDS (blue dashed) and GPFA (red dash-dotted) models. Analyses are based on one movement type with the target in the 45° direction. Cross-prediction error was computed using 4-fold cross-validation. HSLDS with different values of $S$ also outperformed the LDS case (which is equivalent to $S = 1$). Performance was more sensitive to the strength $\psi$ of the Wishart prior, and the best performing model is shown.

trajectories and dynamical transitions estimated by the model predict reaction time, a behavioral covariate that varies from trial-to-trial. Finally we argue that these behavioral correlates are difficult to obtain using a standard neural analysis method.

## 4.1 Cross-prediction

To validate the HSLDS model we compared it to the GPFA model described in section 3.1 and a single LDS model. Since all of these models attempt to capture the shared variance of the data across neurons and multiple trials, we used cross-prediction to measure their performance. Cross-prediction looks at how well the spiking activity of one neuron is predicted just by looking at the spiking activity of all of the other neurons (described in detail in Section 3.2). We found that both the single LDS and HSLDS models that allow for coupled latent dynamics do better than GPFA, shown in Figure 2, which could be attributed to the fact that GPFA constrains the different dimensions of the latent computational state to evolve independently. The HSLDS model also outperforms a single LDS yielding the lowest prediction error for all of the latent dimensions we have looked at, arguing that a nonlinear model of the latent dynamics is better than a linear model. Note that the minimum prediction error asymptotes after 10 latent dimensions. It is tempting to suggest that for this particular task the effective dimensionality of the spiking activity is much lower than that of the 105 recorded neurons, thereby justifying the use of a low-dimensional manifold to describe the underlying computation. This could be interpreted as evidence that neurons may carry redundant information and that the (nonlinear) computational function of the network is better reflected at the level of the population of neurons, rather than in single neurons.

## 4.2 Data segmentation

By definition, the HSLDS model partitions the latent dynamics underlying the observed data into time-labeled segments that may evolve linearly. The segments found by HSLDS correspond to periods of time in which the latent dynamics seem to evolve according to different linear dynamical laws, suggesting that the observed firing pattern of the network has changed as a whole. Thus, by construction, the HSLDS model can subdivide the network activity into different firing regimes for each trial specifically.

For the purpose of visualization, we have applied an additional orthonormalization post-processing step (as in [8]) that helps us order the latent dimensions according to the amount of covariance explained. The orthonormalization consists of finding the singular-value decomposition of $C$, allowing us to write the product $Cx_{:,t}$ as $U_C(D_C V'_C x_{:,t})$, where $U_C \in I\!R_{q \times p}$ is a matrix with orthonormal columns. We will refer to $\tilde{x}_{:,t} = D_C V'_C x_{:,t}$ as the orthonormalised latent state at time $t$. The first dimension of the orthonormalised latent state in time $\tilde{x}_{1,:}$ corresponds then to the latent trajectory which explains the most covariance. Since the columns of $U_C$ are orthonormal, the relationship between the orthonormalised latent trajectories and observed data can be interpreted in an intuitive way, similarly to Principal Components Analysis (PCA). The results presented here were obtained by setting the number of switching LDSs $S$, latent space dimensionality $p$ and Wishart prior $\psi$ to values that yielded a reasonably low cross-prediction error.

Figure 3 shows a typical example of the HSLDS model applied to data in one movement direction, where the different trials are fanned out vertically for illustration purposes. The first orthonormalized

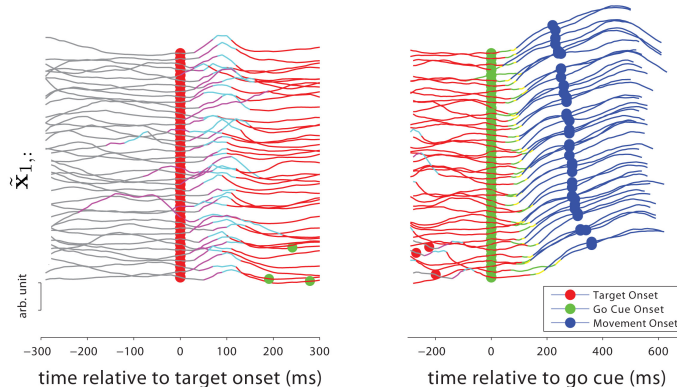

Figure 3: HSLDS applied to neural data from the $45°$ direction movement ($S = 7, p = 7, \psi = 0.05$). The first dimension of the orthonormalised latent trajectory is shown. The colors denote the different linear dynamical systems used by the model. Each line is a different trial, aligned to the target onset (left) and go cue (right), and sorted by reaction time. Switches reliably follow the target onset and precede the movement onset, with a time lag that is correlated with reaction time.

latent dimension indicates a transient in the recorded population activity shortly after target onset (which is marked by the red dots) and a sustained change of activity after the go cue (marked by the green dots). The colours of the lines indicate the most likely setting of the switching variable at each time. It is evident that the learned solution segments each trial into a broadly reproducible sequence of dynamical epochs. Some transitions appear to reliably follow or precede external events (even though these events were not used to train the segmentation) and may reflect actual changes in dynamics due to external influences. Others seem to follow each other in quick succession, and may instead reflect linear approximations to non-linear dynamical processes—evident particularly during transiently rapid changes in the latent state. Unfortunately, the current model does not allow us to distinguish quantitatively between these two types of transition.

Note that the delays (time from target onset to go cue) used in the experiment varied from 200 to 700ms, such that the model systematically detected a change in the neural firing rates shortly after the go cue appeared on each individual trial. The model succeeds at detecting these changes in a purely unsupervised fashion as it was not given any time information about the external experimental inputs.

### 4.3 Behavioral correlates during single trials

It is not surprising that the firing rates of the recorded neurons change during different behavioral periods. For example, neural activity is often observed to be higher during movement execution than during movement preparation. However, the HSLDS method reliably detects the behaviourally-correlated changes in the *pattern* of neural activity across many neurons on *single trials*.

In order to ensure that HSLDS captures trial-specific information we have looked at whether the time post-go-cue at which the model estimates a first switch in the neural dynamics could predict the subsequent onset of movement and thus the trial reaction time (RT). We found that the filtered model (which does not incorporate spiking data from future times into its estimate of the switching variable) could explain 52% of the reaction time variance on average, across the 7 reach directions (Figure 4).

Could a more conventional approach do better? We attempted to use a combination of the "population vector" (PV) method and the "rise-to-threshold" hypothesis. The PV sums the preferred directions of a population of neurons, weighted by the respective spike counts in order to decode the represented direction of movement [22]. The rise-to-threshold hypothesis asserts that neural firing rates rise during a preparatory period and movement is initiated when the population rate crosses a threshold [23]. The neural data used for this analysis were smoothed with a Gaussian window and sampled at 1 ms. We first estimated the preferred direction $\hat{p}_q$ of the neuron indexed by $q$ as the

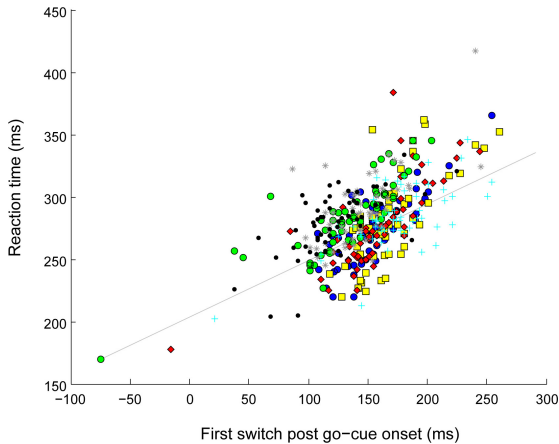

Figure 4: Correlation ($R^2 = 0.52$) between the reaction time and first filtered HSLDS switch following the go cue, on a trial-by-trial basis and averaged across directions. Symbols correspond to movements in different directions. Note that in two catch trials the model did not switch following the go cue, so we considered the last switch before the cue.

unit vector in the direction of $\vec{p}_q = \sum_{d=1}^{7} r_i^d \vec{v}^d$ where $d$ indexes the instructed movement direction $\vec{v}^d$ and $r_q^d$ is the mean firing rate of neuron $q$ during all movements in direction $d$. The preferred direction of a given neuron often differed between plan and movement activity, so we used data from movement onset until the movement end to estimate $r_q^d$ as this gave us better results when trying to estimate a threshold in the rising movement-related activity. We then estimated the instanteneous amplitude of the network PV at time t as $s_t^d = || \sum_{q=1}^{Q} y_{q,t} \vec{p}_q ||$, where $y_{q,t}$ is the smoothed spike count of neuron $q$ at time $t$, $Q$ is the number of neurons and $||\vec{w}||$ denotes the norm of the vector $\vec{w}$. Finally, we searched for a threshold length (one per direction), such that the time at which the PV exceeded this length on each trial was best correlated with RT.

Note that this approach uses considerable supervision that was denied to the HSLDS model. First, the movement epoch of each trial was identified to define the PV. Second, the thresholds were selected so as to maximize the RT correlation—a direct form of supervision. Finally, this selection was based on the same data as were used to evaluate the correlation score, thus leading to potential overfitting in the explained variance. The HSLDS model was also trained on the same trials, which could lead to some overfitting in terms of likelihood, but should not introduce overfitting in the correlation between switch times and RT, which is not directly optimised.

Despite these considerable advantages, the PV approach did not predict RT as well as did the HSLDS, yielding an average variance explained across conditions of 48%.

## 5  Conclusion

It appears that the Hidden Switching Linear Dynamical System (HSLDS) model is able to appropriately extract relevant aspects of the computation reflected in a network of firing neurons. HSLDS explicitly models the nonlinear dynamics of the computation as a piecewise linear process that captures the shared variance in the neural data across neurons and multiple trials.

One limitation of HSLDS is the approximate EM algorithm used for inference and learning of the model parameters. We have traded off computational tractability with accuracy, such that the model may settle into a solution that is simpler than the optimum. A second limitation of HSLDS is the slow training time of EM, enforcing an offline learning of the model parameters.

Despite these simplications, HSLDS can be used to dynamically segment the neural activity at the level of the whole population of neurons into periods of different firing regimes. We showed that in a delayed-reach task the firing regimes found correlate well with the experimental behavioral periods. The computational trajectories found by HSLDS are trial-specific and with a dimensionality that is more suitable for visualization than the high-dimensional spiking activity. Overall, HSLDS are attractive models for uncovering behavioral correlates in neural data on a single-trial basis.

**Acknowledgments.**   This work was supported by DARPA REPAIR (N66001-10-C-2010), the Swiss National Science Foundation Fellowship PBELP3-130908, the Gatsby Charitable Foundation, UK EPSRC EP/H019472/1 and NIH-NINDS-CRCNS-R01, NDSEG and NSF Graduate Fellowships, Christopher and Dana Reeve Foundation. We are very grateful to Jacob Macke, Lars Buesing and Alexander Lerchner for discussion.

# References

[1] A. C. Smith and E. N. Brown. Estimating a state-space model from point process observations. *Neural Computation*, 15(5):965–991, 2003.

[2] M. Stopfer, V. Jayaraman, and G. Laurent. Intensity versus identity coding in an olfactory system. *Neuron*, 39:991–1004, 2003.

[3] S. L. Brown, J. Joseph, and M. Stopfer. Encoding a temporally structured stimulus with a temporally structured neural representation. *Nature Neuroscience*, 8(11):1568–1576, 2005.

[4] R. Levi, R. Varona, Y. I. Arshavsky, M. I. Rabinovich, and A. I. Selverston. The role of sensory network dynamics in generating a motor program. *Journal of Neuroscience*, 25(42):9807–9815, 2005.

[5] O. Mazor and G. Laurent. Transient dynamics versus fixed points in odor representations by locust antennal lobe projection neurons. *Neuron*, 48:661–673, 2005.

[6] B. M. Broome, V. Jayaraman, and G. Laurent. Encoding and decoding of overlapping odor sequences. *Neuron*, 51:467–482, 2006.

[7] M. M. Churchland, B. M. Yu, M. Sahani, and K. V. Shenoy. Techniques for extracting single-trial activity patterns from large-scale neural recordings. *Current Opinion in Neurobiology*, 17(5):609–618, 2007.

[8] B. M. Yu, J. P. Cunningham, G. Santhanam, S. I. Ryu, K. V. Shenoy, and M. Sahani. Gaussian-process factor analysis for low-dimensional single-trial analysis of neural population activity. *J Neurophysiol*, 102:614–635, 2009.

[9] Y. Bar-Shalom and Xiao-Rong Li. *Estimation and Tracking: Principles, Techniques and Software*. Artech House, Norwood, MA, 1998.

[10] B. Mesot and D. Barber. Switching linear dynamical systems for noise robust speech recognition. *IEEE Transactions of Audio, Speech and Language Processing*, 15(6):1850–1858, 2007.

[11] W. Wu, M.J. Black, D. Mumford, Y. Gao, E. Bienenstock, and J. P. Donoghue. Modeling and decoding motor cortical activity using a switching kalman filter. *IEEE Transactions on Biomedical Engineering*, 51(6):933–942, 2004.

[12] D. Barber. *Bayesian Reasoning and Machine Learning*. Cambridge University Press. In Press, 2011.

[13] K. P. Murphy. Switching kalman filters. Technical Report 98-10, Compaq Cambridge Research Lab, 1998.

[14] B. M. Yu, A. Afshar, G. Santhanam, S. I. Ryu, K. V. Shenoy, and M. Sahani. Extracting dynamical structure embedded in neural activity. In Y. Weiss, B. Schölkopf, and J. Platt, editors, *Advances in Neural Information Processing Systems 18*, pages 1545–1552. Cambridge, MA: MIT Press, 2006.

[15] M. West and J. Harrison. *Bayesian Forecasting and Dynamic Models*. Springer, 1999.

[16] D. L. Alspach and H. W. Sorenson. Nonlinear bayesian estimation using gaussian sum approximations. *IEEE Transactions on Automatic Control*, 17(4):439–448, 1972.

[17] X. Boyen and D. Koller. Tractable inference for complex stochastic processes. In *Proceedings of the 14th Conference on Uncertainty in Artificial Intelligence - UAI*, volume 17, pages 33–42. Morgan Kaufmann, 1998.

[18] T. Minka. *A Family of Algorithms for Approximate Bayesian Inference*. PhD Thesis, MIT Media Lab, 2001.

[19] D. Barber. Expectation correction for smoothed inference in switching linear dynamical systems. *Journal of Machine Learning Research*, 7:2515–2540, 2006.

[20] A. J. Viterbi. Error bounds for convolutional codes and an asymptotically optimum decoding algorithm. *IEEE Transactions on Information Theory*, IT-13:260–267, 1967.

[21] N. A. Thacker and P. A. Bromiley. The effects of a square root transform on a poisson distributed quantity. Technical Report 2001-010, University of Manchester, 2001.

[22] A. P. Georgopoulos, A. B. Schwartz, and R. E. Ketiner. Neuronal population coding of movement direction. *Science*, 233:1416–1419, 1986.

[23] W. Erlhagen and G. Schoner. Dynamic field theory of movement preparation. *Psychol Rev*, 109:545–572, 2002.

